# Unsupervised Learning of Mixtures of Multiple Causes in Binary Data

**Eric Saund**
Xerox Palo Alto Research Center
3333 Coyote Hill Rd., Palo Alto, CA, 94304

## Abstract

This paper presents a formulation for unsupervised learning of clusters reflecting multiple causal structure in binary data. Unlike the standard mixture model, a multiple cause model accounts for observed data by combining assertions from many hidden causes, each of which can pertain to varying degree to any subset of the observable dimensions. A crucial issue is the *mixing-function* for combining beliefs from different cluster-centers in order to generate data reconstructions whose errors are minimized both during recognition and learning. We demonstrate a weakness inherent to the popular weighted sum followed by sigmoid squashing, and offer an alternative form of the nonlinearity. Results are presented demonstrating the algorithm's ability successfully to discover coherent multiple causal representations of noisy test data and in images of printed characters.

## 1 Introduction

The objective of unsupervised learning is to identify patterns or features reflecting underlying regularities in data. *Single-cause* techniques, including the k-means algorithm and the standard mixture-model (Duda and Hart, 1973), represent clusters of data points sharing similar patterns of 1s and 0s under the assumption that each data point belongs to, or was generated by, one and only one cluster-center; output activity is constrained to sum to 1. In contrast, a *multiple-cause* model permits more than one cluster-center to become fully active in accounting for an observed data vector. The advantage of a multiple cause model is that a relatively small number

of hidden variables can be applied combinatorially to generate a large data set. Figure 1 illustrates with a test set of nine 121-dimensional data vectors. This data set reflects two independent processes, one of which controls the position of the black square on the left hand side, the other controlling the right. While a single cause model requires nine cluster-centers to account for this data, a perspicuous multiple cause formulation requires only six hidden units as shown in figure 4b. Grey levels indicate dimensions for which a cluster-center adopts a "don't-know/don't-care" assertion.

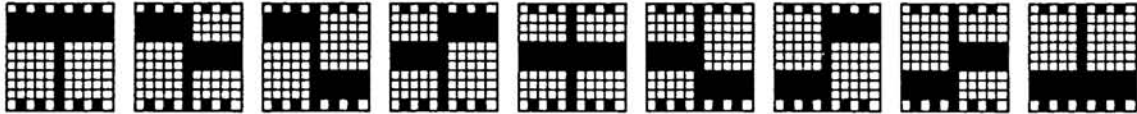

Figure 1: Nine 121-dimensional test data samples exhibiting multiple cause structure. Independent processes control the position of the black rectangle on the left and right hand sides.

While principal components analysis and its neural-network variants (Bourlard and Kamp, 1988; Sanger, 1989) as well as the Harmonium Boltzmann Machine (Freund and Haussler, 1992) are inherently multiple cause models, the hidden representations they arrive at are for many purposes intuitively unsatisfactory. Figure 2 illustrates the principal components representation for the test data set presented in figure 1. Principal components is able to reconstruct the data without error using only four hidden units (plus fixed centroid), but these vectors obscure the compositional structure of the data in that they reveal nothing about the statistical independence of the left and right hand processes. Similar results obtain for multiple cause unsupervised learning using a Harmonium network and for a feedforward network using the sigmoid nonlinearity. We seek instead a multiple cause formulation which will deliver *coherent* representations exploiting "don't-know/don't-care" weights to make explicit the statistical dependencies *and* independencies present when clusters occur in lower-dimensional subspaces of the full $J$-dimensional data space.

Data domains differ in ways that underlying causal processes interact. The present discussion focuses on data obeying a WRITE-WHITE-AND-BLACK model, under which hidden causes are responsible for both turning "on" and turning "off" the observed variables.

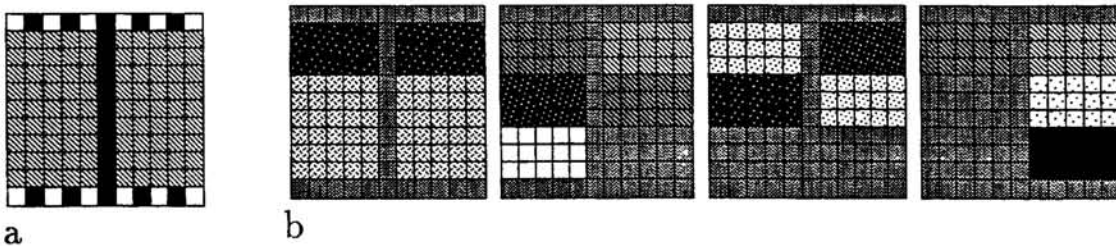

a                    b

Figure 2: Principal components representation for the test data from figure 1. (a) centroid (white: $-1$, black: $1$). (b) four component vectors sufficient to encode the nine data points. (lighter shadings: $c_{j,k} < 0$; grey: $c_{j,k} = 0$; darker shading: $c_{j,k} > 0$).

## 2   Mixing Functions

A large class of unsupervised learning models share the architecture shown in figure 3. A binary vector $D_i \equiv (d_{i,1}, d_{i,2}, ...d_{i,j}, ...d_{i,J})$ is presented at the data layer, and a *measurement*, or response vector $m_i \equiv (m_{i,1}, m_{i,2}, ...m_{i,k}, ...m_{i,K})$ is computed at the encoding layer using "weights" $c_{j,k}$ associating activity at data dimension $j$ with activity at hidden cluster-center $k$. Any activity pattern at the encoding layer can be turned around to compute a *prediction* vector $r_i \equiv (r_{i,1}, r_{i,2}, ...r_{i,j}, ...r_{i,J})$ at the data layer. Different models employ different functions for performing the measurement and prediction mappings, and give different interpretations to the weights. Common to most models is a learning procedure which attempts to optimize an objective function on errors between data vectors in a training set, and predictions of these data vectors under their respective responses at the encoding layer.

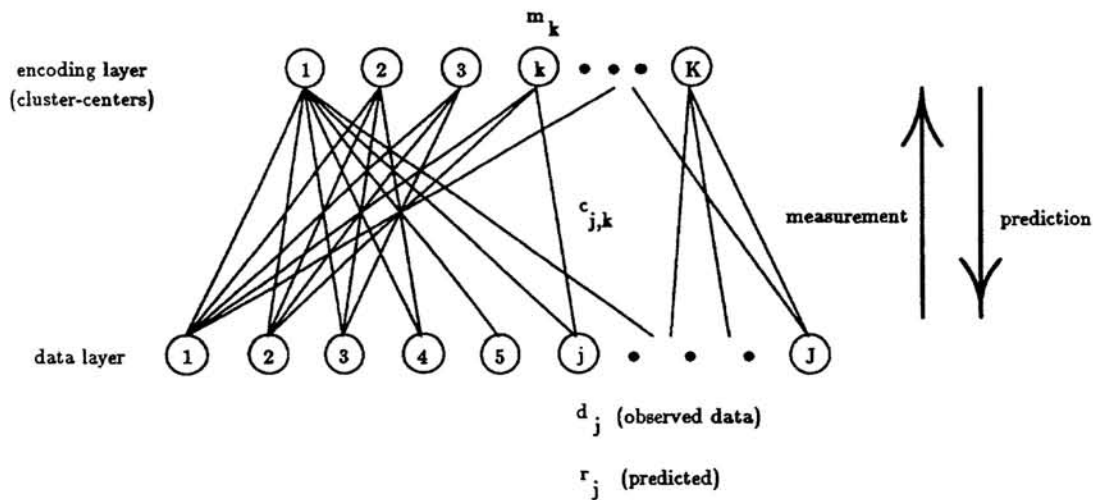

Figure 3: Architecture underlying a large class of unsupervised learning models.

The key issue is the *mixing function* which specifies how sometimes conflicting predictions from individual hidden units combine to predict values on the data dimensions. Most neural-network formulations, including principal components variants and the Boltzmann Machine, employ linearly weighted sum of hidden unit activity *followed* by a squashing, bump, or other nonlinearity. This form of mixing function permits an error in prediction by one cluster center to be cancelled out by correct predictions from others without consequence in terms of error in the net prediction. As a result, there is little global pressure for cluster-centers to adopt don't-know values when they are not quite confident in their predictions.

Instead, a multiple cause formulation delivering coherent cluster-centers requires a form of nonlinearity in which active disagreement must result in a net "uncertain" or neutral prediction that results in nonzero error.

## 3    Multiple Cause Mixture Model

Our formulation employs a *zero-based* representation at the data layer to simplify the mathematical expression for a suitable mixing function. Data values are either 1 or $-1$; the sign of a weight $c_{j,k}$ indicates whether activity in cluster-center $k$ predicts a 1 or $-1$ at data dimension $j$, and its magnitude ($|c_{j,k}| \leq 1$) indicates strength of belief; $c_{j,k} = 0$ corresponds to "don't-know/don't-care" (grey in figure 4b).

The mixing function takes the form,

$$r_{i,j} = \frac{\left[\sum_{k\ c_{j,k}<0} m_{i,k}(-c_{j,k})\right]\left[\prod_{k\ c_{j,k}<0}(1+m_{i,k}c_{j,k})-1\right]+\left[\sum_{k\ c_{j,k}>0} m_{i,k}c_{j,k}\right]\left[1-\prod_{k\ c_{j,k}>0}(1-m_{i,k}c_{j,k})\right]}{\sum_k m_{i,k}|c_{j,k}|}$$

This formula is a computationally tractable approximation to an idealized mixing function created by linearly interpolating boundary values on the extremes of $m_{i,k} \in \{0, 1\}$ and $c_{j,k} \in \{-1, 0, 1\}$ rationally designed to meet the criteria outlined above.

Both learning and measurement operate in the context of an objective function on predictions equivalent to log-likelihood. The weights $c_{j,k}$ are found through gradient ascent in this objective function, and at each training step the encoding $m_i$ of an observed data vector is found by gradient ascent as well.

## 4    Experimental Results

Figure 4 shows that the model converges to the coherent multiple cause representation for the test data of figure 1 starting with random initial weights. The model is robust with respect to noisy training data as indicated in figure 5.

In figure 6 the model was trained on data consisting of $21 \times 21$ pixel images of registered lower case characters. Results for $K = 14$ are shown indicating that the model has discovered statistical regularities associated with ascenders, descenders, circles, etc.

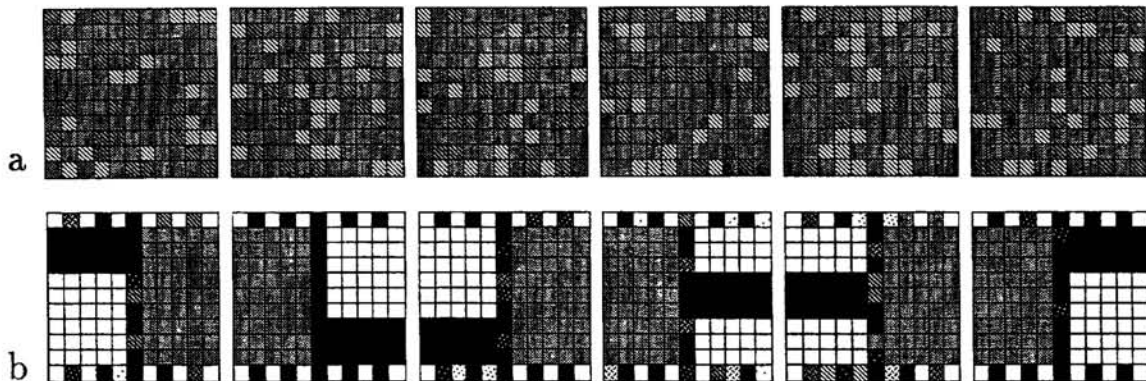

Figure 4: Multiple Cause Mixture Model representation for the test data from figure 1. (a) Initial random cluster-centers. (b) Cluster-centers after seven training iterations (white: $c_{j,k} = -1$; grey: $c_{j,k} = 0$; black: $c_{j,k} = 1$).

# 5  Conclusion

Ability to compress data, and statistical independence of response activities (Barlow, 1989), are not the only criteria by which to judge the success of an encoder network paradigm for unsupervised learning. For many purposes, it is equally important that hidden units make explicit statistically salient structure arising from causally distinct processes.

The difficulty lies in getting the internal knowledge-bearing entities sensibly to divvy up responsibility for training data not just pointwise, but dimensionwise. Mixing functions based on linear weighted sum of activities (possibly followed by a nonlinearity) fail to achieve this because they fail to pressure the hidden units into giving up responsibility (adopting "don't know" values) for data dimensions on which they are prone to be incorrect. We have outlined criteria, and offered a specific functional form, for nonlinearly combining beliefs in a predictive mixing function such that statistically coherent hidden representations of multiple causal structure can indeed be discovered in binary data.

## References

Barlow, H.; [1989], "Unsupervised Learning," *Neural Computation*, 1: 295-311.

Bourlard, H., and Kamp, Y.; [1988], Auto-Association by Multilayer Perceptrons and Singular Value Decomposition," *Biological Cybernetics*, 59:4-5, 291-294.

Duda, R., and Hart, P.; [1973], *Pattern Classification and Scene Analysis*, Wiley, New York.

Földiák, P.; [1990], "Forming sparse representations by local anti-Hebbian learning," *Biological Cybernetics*, 64:2, 165-170.

Freund, Y., and Haussler, D.; [1992], "Unsupervised learning of distributions on binary vectors using two-layer networks," in Moody, J., Hanson, S., and Lippman, R., eds, *Advances in Neural Information Processing Systems 4*, Morgan Kauffman, San Mateo, 912-919.

Nowlan, S.; [1990], "Maximum Likelihood Competitive Learning," in Touretzky, D., ed., *Advances in Neural Information Processing Systems 2*, Morgan Kauffman, San Mateo, 574-582.

Sanger, T.; [1989], "An Optimality Principle for Unsupervised Learning," in Touretzky, D., ed., *Advances in Neural Information Processing Systems*, Morgan Kauffman, San Mateo, 11-19.

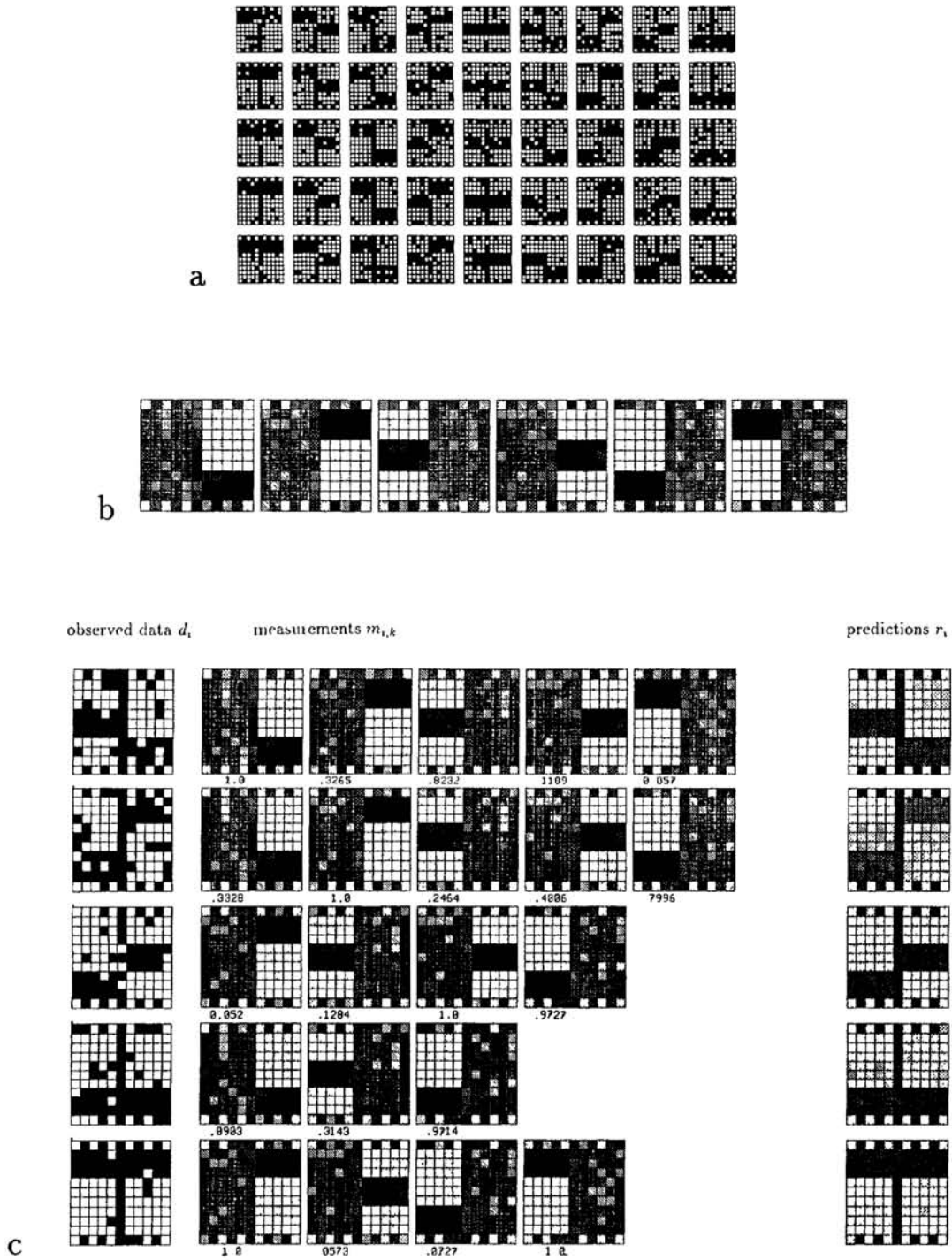

Figure 5: Multiple Cause Mixture Model results for noisy training data. (a) Five test data sample suites with 10% bit-flip noise. Twenty suites were used to train from random initial cluster-centers, resulting in the representation shown in (b). (c) Left: Five test data samples $d_i$; Middle: Numerical activities $m_{i,k}$ for the most active cluster-centers (the corresponding cluster-center is displayed above each $m_{i,k}$ value); Right: reconstructions (predictions) $r_i$ based on the activities. Note how these "clean up" the noisy samples from which they were computed.

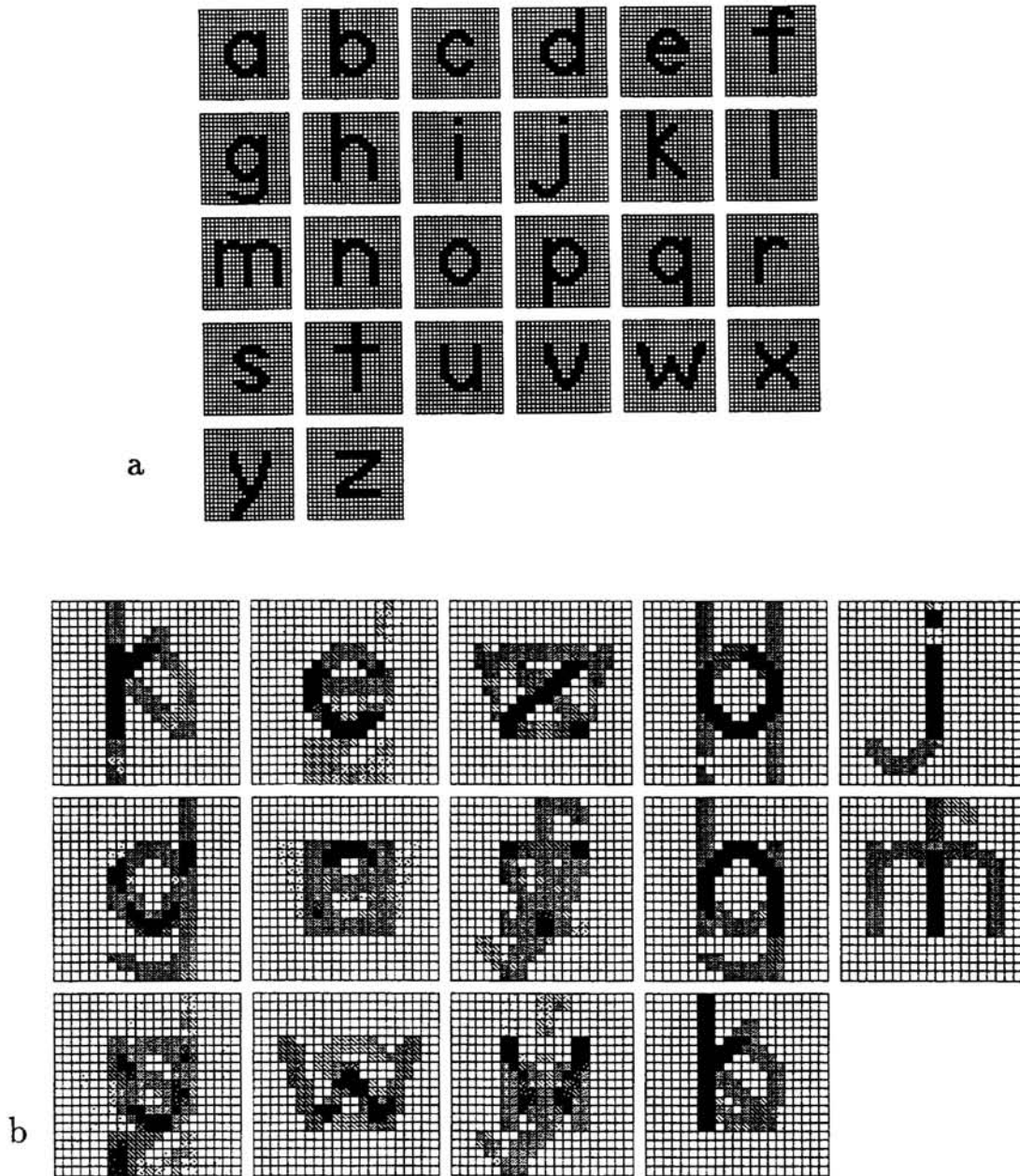

Figure 6: (a) Training set of twenty-six 441-dimensional binary vectors. (b) Multiple Cause Mixture Model representation at $K = 14$. (c) Left: Five test data samples $d_i$; Middle: Numerical activities $m_{i,k}$ for the most active cluster-centers (the corresponding cluster-center is displayed above each $m_{i,k}$ value); Right: reconstructions (predictions) $r_i$ based on the activities.

observed data $d_i$          measurements $m_{i,k}$                                    predictions $r_i$

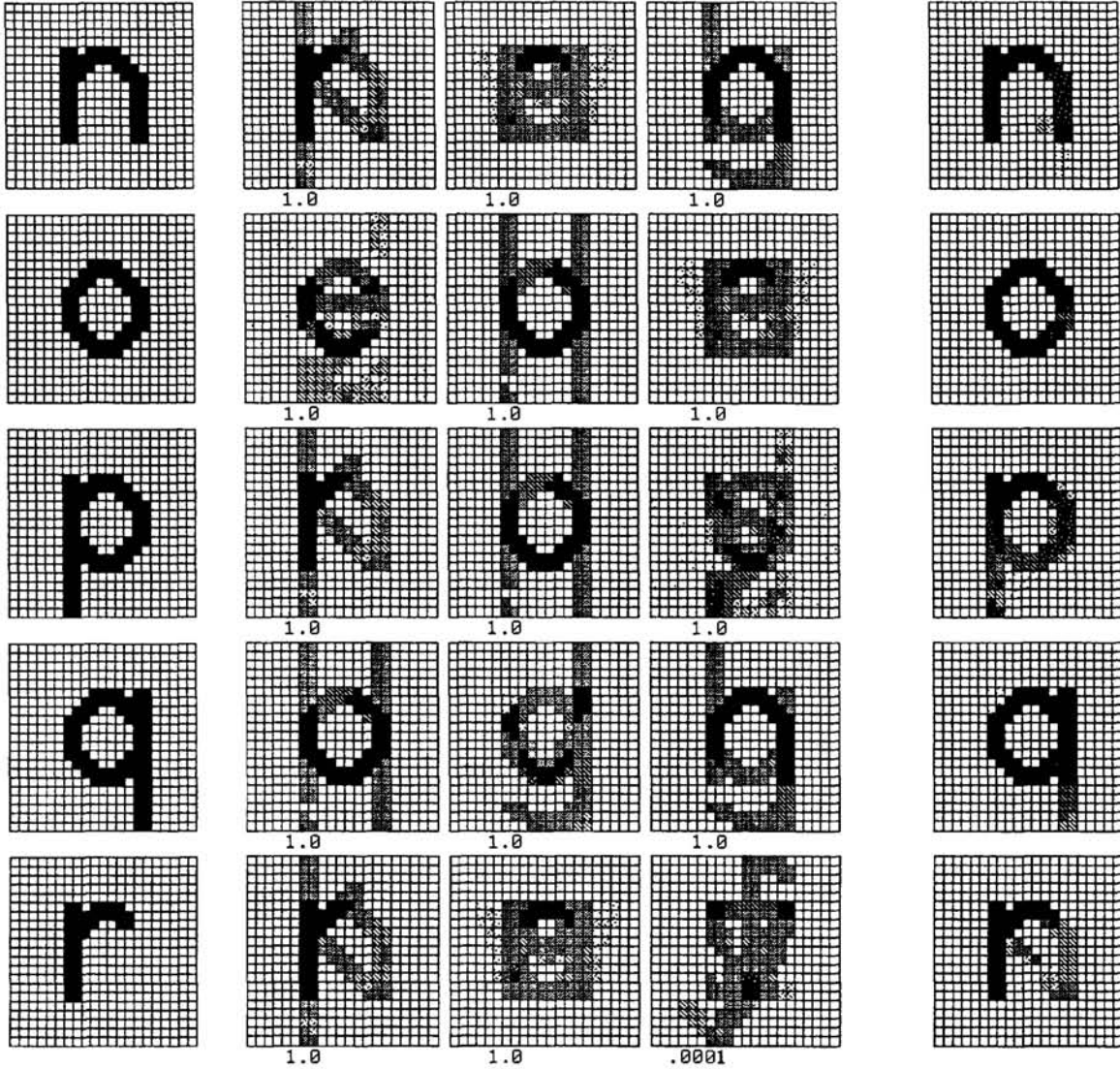

c